# Using Manifold Structure for Partially Labelled Classification

**Mikhail Belkin**
University of Chicago
Department of Mathematics
misha@math.uchicago.edu

**Partha Niyogi**
University of Chicago
Depts of Computer Science and Statistics
niyogi@cs.uchicago.edu

## Abstract

We consider the general problem of utilizing both labeled and un-labeled data to improve classification accuracy. Under the assumption that the data lie on a submanifold in a high dimensional space, we develop an algorithmic framework to classify a partially labeled data set in a principled manner. The central idea of our approach is that classification functions are naturally defined only on the submanifold in question rather than the total ambient space. Using the Laplace Beltrami operator one produces a basis for a Hilbert space of square integrable functions on the submanifold. To recover such a basis, only unlabeled examples are required. Once a basis is obtained, training can be performed using the labeled data set. Our algorithm models the manifold using the adjacency graph for the data and approximates the Laplace Beltrami operator by the graph Laplacian. Practical applications to image and text classification are considered.

## 1  Introduction

In many practical applications of data classification and data mining, one finds a wealth of easily available unlabeled examples, while collecting labeled examples can be costly and time-consuming. Standard examples include object recognition in images, speech recognition, classifying news articles by topic. In recent times, genetics has also provided enormous amounts of readily accessible data. However, classification of this data involves experimentation and can be very resource intensive. Consequently it is of interest to develop algorithms that are able to utilize both labeled and unlabeled data for classification and other purposes. Although the area of partially labeled classification is fairly new, a considerable amount of work has been done in that field since the early 90's, see [2, 4, 7]. In this paper we address the problem of classifying a partially labeled set by developing the ideas proposed in [1] for data representation. In particular, we exploit the intrinsic structure of the data to improve classification with unlabeled examples under the assumption

that the data resides on a low-dimensional manifold within a high-dimensional representation space. In some cases it seems to be a reasonable assumption that the data lies on or close to a manifold. For example a handwritten digit **0** can be fairly accurately represented as an ellipse, which is completely determined by the coordinates of its foci and the sum of the distances from the foci to any point. Thus the space of ellipses is a five-dimensional manifold. An actual handwritten **0** would require more parameters, but perhaps not more than 15 or 20. On the other hand the dimensionality of the ambient representation space is the number of pixels which is typically far higher. For other types of data the question of the manifold structure seems significantly more involved. While there has been recent work on using manifold structure for data representation ([6, 8]), the only other application to classification problems that we are aware of, was in [7], where the authors use a random walk on the data adjacency graph for partially labeled classification.

## 2 Why Manifold Structure is Useful for Partially Supervised Learning

To provide a motivation for using a manifold structure, consider a simple synthetic example shown in Figure 1. The two classes consist of two parts of the curve shown in the first panel (row 1). We are given a few labeled points and 500 unlabeled points shown in panels 2 and 3 respectively. The goal is to establish the identity of the point labeled with a question mark. By observing the picture in panel 2 (row 1) we see that we cannot confidently classify "?" by using the labeled examples alone. On the other hand, the problem seems much more feasible given the unlabeled data shown in panel 3. Since there is an underlying manifold, it seems clear at the outset that the (geodesic) distances along the curve are more meaningful than Euclidean distances in the plane. Therefore rather than building classifiers defined on the plane ($\mathbb{R}^2$) it seems preferable to have classifiers defined on the curve itself. Even though the data has an underlying manifold, the problem is still not quite trivial since the two different parts of the curve come confusingly close to each other. There are many possible potential representations of the manifold and the one provided by the curve itself is unsatisfactory. Ideally, we would like to have a representation of the data which captures the fact that it is a closed curve. More specifically, we would like an embedding of the curve where the coordinates vary as slowly as possible when one traverses the curve. Such an ideal representation is shown in the panel 4 (first panel of the second row). Note that both represent the same underlying manifold structure but with different coordinate functions. It turns out (panel 6) that by taking a two-dimensional representation of the data with Laplacian Eigenmaps [1], we get very close to the desired embedding. Panel 5 shows the locations of labeled points in the new representation space. We see that "?" now falls squarely in the middle of "+" signs and can easily be identified as a "+".

This artificial example illustrates that recovering the manifold and developing classifiers on the manifold itself might give us an advantage in classification problems. To recover the manifold, all we need is unlabeled data. The labeled data is then used to develop a classifier defined on this manifold. However we need a model for the manifold to utilize this structure. The model used here is that of a weighted graph whose vertices are data points. Two data points are connected with an edge if

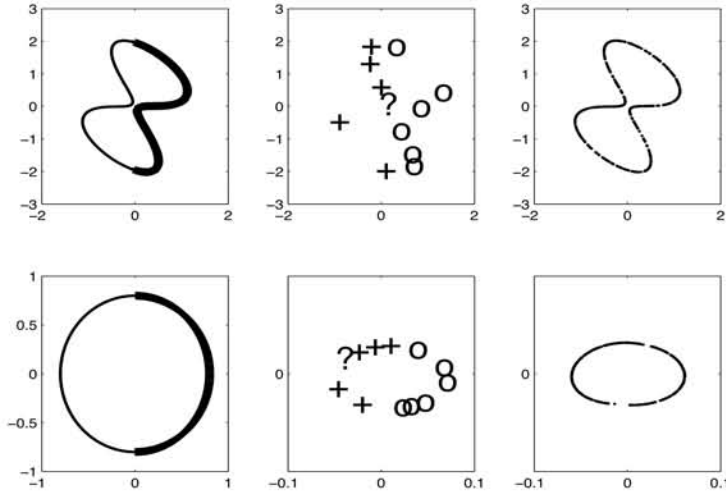

Figure 1: Top row: Panel 1. Two classes on a plane curve. Panel 2. Labeled examples. "?" is a point to be classified. Panel 3. 500 random unlabeled examples. Bottom row: Panel 4. Ideal representation of the curve. Panel 5. Positions of labeled points and "?" after applying eigenfunctions of the Laplacian. Panel 6. Positions of all examples.

and only if the points are sufficiently close. To each edge we can associate a distance between the corresponding points. The "geodesic distance" between two vertices is the length of the shortest path between them on the adjacency graph. Once we set up an approximation to the manifold, we need a method to exploit the structure of the model to build a classifier. One possible simple approach would be to use the "geodesic nearest neighbors". However, while simple and well-motivated, this method is potentially unstable. A related more sophisticated method based on a random walk on the adjacency graph is proposed in [7]. We also note the approach taken in [2] which uses mincuts of certain graphs for partially labeled classifications.

Our approach is based on the Laplace-Beltrami operator defined on Riemannian manifolds (see [5]). The eigenfunctions of the Laplace Beltrami operator provide a natural basis for functions on the manifold and the desired classification function can be expressed in such a basis. The Laplace Beltrami operator can be estimated using unlabeled examples alone and the classification function is then approximated using the labeled data. In the next two sections we describe our algorithm and the theoretical underpinnings in some detail.

## 3  Description of the Algorithm

Given $k$ points $\mathbf{x}_1, \ldots, \mathbf{x}_k \in \mathbb{R}^l$, we assume that the first $s < k$ points have labels $c_i$, where $c_i \in \{-1, 1\}$ and the rest are unlabeled. The goal is to label the unlabeled points. We also introduce a straightforward extension of the algorithm for the case of more than two classes.

Step 1 [*Constructing the Adjacency Graph with n nearest neighbors*]. Nodes $i$ and

$j$ corresponding to the points $\mathbf{x}_i$ and $\mathbf{x}_j$ are connected by an edge if $i$ is among $n$ nearest neighbors of $j$ or $j$ is among $n$ nearest neighbors of $i$. The distance can be the standard Euclidean distance in $\mathbb{R}^l$ or some other appropriately defined distance. We take $w_{ij} = 1$ if points $\mathbf{x}_i$ and $\mathbf{x}_j$ are connected and $w_{ij} = 0$ otherwise. For a discussion about the appropriate choice of weights, and connections to the heat kernel see [1].

Step 2. [*Eigenfunctions*] Compute $p$ eigenvectors $\mathbf{e}_1, \ldots, \mathbf{e}_p$ corresponding to the $p$ smallest eigenvalues for the eigenvector problem $L\mathbf{e} = \lambda \mathbf{e}$ where $L = D - W$ is the graph Laplacian for the adjacency graph. Here $W$ is the adjacency matrix defined above and $D$ is a diagonal matrix of the same size as $W$ satisfying $D_{ii} = \sum_j W_{ij}$. Laplacian is a symmetric, positive semidefinite matrix which can be thought of as an operator on functions defined on vertices of the graph.

Step 3. [*Building the classifier*] To approximate the class we minimize the error function $\mathrm{Err}(\mathbf{a}) = \sum_{i=1}^{s} \left( c_i - \sum_{j=1}^{p} a_j \mathbf{e}_j(i) \right)^2$ where $p$ is the number of eigenfunctions we wish to employ, the sum is taken over all labeled points and the minimization is considered over the space of coefficients $\mathbf{a} = (a_1, \ldots, a_p)^T$. The solution is given by

$$\mathbf{a} = \left( \mathbf{E}_{\mathrm{lab}}^T \mathbf{E}_{\mathrm{lab}} \right)^{-1} \mathbf{E}_{\mathrm{lab}}^T \mathbf{c}$$

where $\mathbf{c} = (c_1, \ldots, c_s)^T$ and $\mathbf{E}_{\mathrm{lab}}$ is an $s \times p$ matrix whose $i, j$ entry is $\mathbf{e}_j(i)$. For the case of several classes, we build a one-against-all classifier for each individual class.

Step 4. [*Classifying unlabeled points*] If $\mathbf{x}_i$, $i > s$ is an unlabeled point we put

$$c_i = \begin{cases} 1, & \text{if } \sum_{j=1}^{p} a_j \mathbf{e}_j(i) \geq 0 \\ -1, & \text{if } \sum_{j=1}^{p} a_j \mathbf{e}_j(i) < 0 \end{cases}$$

This, of course, is just applying a linear classifier constructed in Step 3. If there are several classes, one-against-all classifiers compete using $\sum_{j=1}^{p} a_j \mathbf{e}_j(i)$ as a confidence measure.

## 4    Theoretical Interpretation

Let $\mathcal{M} \subset \mathbb{R}^k$ be an $n$-dimensional compact Riemannian manifold isometrically embedded in $\mathbb{R}^k$ for some $k$. Intuitively $\mathcal{M}$ can be thought of as an $n$-dimensional "surface" in $\mathbb{R}^k$. Riemannian structure on $\mathcal{M}$ induces a volume form that allows us to integrate functions defined on $\mathcal{M}$. The square integrable functions form a Hilbert space $\mathcal{L}^2(\mathcal{M})$. The Laplace-Beltrami operator $\Delta_{\mathcal{M}}$ (or just $\Delta$) acts on twice differentiable functions on $\mathcal{M}$. There are three important points that are relevant to our discussion here.

**The Laplacian provides a basis on $\mathcal{L}^2(\mathcal{M})$:**
It can be shown (e.g., [5]) that $\Delta$ is a self-adjoint positive semidefinite operator and that its eigenfunctions form a basis for the Hilbert space $\mathcal{L}^2(\mathcal{M})$. The spectrum of $\Delta$ is discrete (provided $\mathcal{M}$ is compact) , with the smallest eigenvalue 0 corresponding to the constant eigenfunction. Therefore any $f \in \mathcal{L}^2(\mathcal{M})$ can be written as $f(\mathbf{x}) = \sum_{i=0}^{\infty} a_i e_i(\mathbf{x})$, where $e_i$ are eigenfunctions, $\Delta e_i = \lambda_i e_i$.

The simplest nontrivial example is a circle $S^1$. $\Delta_{S^1} f(\phi) = -\frac{d^2 f(\phi)}{d\phi^2}$. Therefore

the eigenfunctions are given by $-\frac{d^2 e(\phi)}{d\phi^2} = e(\phi)$, where $f(\phi)$ is a $\pi$-periodic function. It is easy to see that all eigenfunctions of $\Delta$ are of the form $e(\phi) = \sin(n\phi)$ or $e(\phi) = \cos(n\phi)$ with eigenvalues $\{1^2, 2^2, \ldots\}$. Therefore, we see that any $\pi$-periodic $\mathcal{L}^2$ function $f$ has a convergent Fourier series expansion given by $f(\phi) = \sum_{n=0}^{\infty} a_n \sin(n\phi) + b_n \cos(n\phi)$. In general, for any manifold $\mathcal{M}$, the eigenfunctions of the Laplace-Beltrami operator provide a natural basis for $\mathcal{L}^2(\mathcal{M})$. However $\Delta$ provides more than just a basis, it also yields a measure of smoothness for functions on the manifold.

**The Laplacian as a smoothness functional**:
A simple measure of the degree of smoothness for a function $f$ on a unit circle $S^1$ is the "smoothness functional" $\mathcal{S}(f) = \int_{S^1} |f(\phi)'|^2 d\phi$. If $\mathcal{S}(f)$ is close to zero, we think of $f$ as being "smooth". Naturally, constant functions are the most "smooth". Integration by parts yields $\mathcal{S}(f) = \int_{S^1} f'(\phi) df = \int_{S^1} f \Delta f d\phi = \langle \Delta f, f \rangle_{\mathcal{L}^2(S^1)}$. In general, if $f : \mathcal{M} \to \mathbb{R}$, then

$$\mathcal{S}(f) \stackrel{\text{def}}{=} \int_{\mathcal{M}} |\nabla f|^2 d\mu = \int_{\mathcal{M}} f \Delta f d\mu = \langle \Delta f, f \rangle_{\mathcal{L}^2(\mathcal{M})}$$

where $\nabla f$ is the gradient vector field of $f$. If the manifold is $\mathbb{R}^n$ then $\nabla f = \sum_{i=1}^{n} \frac{\partial f}{\partial x_i} \frac{\partial}{\partial x_i}$. In general, for an $n$-manifold, the expression in a local coordinate chart involves the coefficients of the metric tensor. Therefore the smoothness of a unit norm eigenfunction $e_i$ of $\Delta$ is controlled by the corresponding eigenvalue $\lambda_i$ since $\mathcal{S}(e_i) = \langle \Delta e_i, e_i \rangle_{\mathcal{L}^2(\mathcal{M})} = \lambda_i$. For an arbitrary $f = \sum_i \alpha_i e_i$, we can write $\mathcal{S}(f)$ as

$$\mathcal{S}(f) = \langle \Delta f, f \rangle = \langle \sum_i \alpha_i \Delta e_i, \sum_i \alpha_i e_i \rangle = \sum_i \lambda_i \alpha_i^2$$

A Reproducing Kernel Hilbert Space can be constructed from $\mathcal{S}$. $\lambda_1 = 0$ is the smallest eigenvalue for which the corresponding eigenfunction is the constant function $e_1 = \frac{1}{\mu(\mathcal{M})}$. It can also be shown that if $\mathcal{M}$ is compact and connected there are no other eigenfunctions with eigenvalue 0. Therefore approximating a function $f(x) \approx \sum_1^p a_i e_i(x)$ in terms of the first $p$ eigenfunctions of $\Delta$ is a way of controlling the smoothness of the approximation. The optimal approximation is obtained by minimizing the $\mathcal{L}^2$ norm of the error: $\mathbf{a} = \underset{\mathbf{a}=(a_1,\ldots,a_p)}{\text{argmin}} \int_{\mathcal{M}} \left( f(\mathbf{x}) - \sum_i^p a_i e_i(\mathbf{x}) \right)^2 d\mu$.
This approximation is given by a projection in $\mathcal{L}^2$ onto the span of the first $p$ eigenfunctions $a_i = \int_{\mathcal{M}} e_i(\mathbf{x}) f(\mathbf{x}) d\mu = \langle e_i, f \rangle_{\mathcal{L}^2(\mathcal{M})}$ In practice we only know the values of $f$ at a finite number of points $\mathbf{x}_1, \cdots, \mathbf{x}_n$ and therefore have to solve a discrete version of this problem $\bar{\mathbf{a}} = \underset{\bar{\mathbf{a}}=(\bar{a}_1,\ldots,\bar{a}_p)}{\text{argmin}} \sum_{i=1}^{n} \left( f(\mathbf{x}_i) - \sum_{j=1}^{p} \bar{a}_j e_j(\mathbf{x}_i) \right)^2$ The solution to this standard least squares problem is given by $\bar{\mathbf{a}}^T = (\mathbf{E}^T \mathbf{E})^{-1} \mathbf{E} \mathbf{y}^T$, where $\mathbf{E}_{ij} = e_i(\mathbf{x}_j)$ and $\mathbf{y} = (f(\mathbf{x}_1), \ldots, f(\mathbf{x}_n))$.

**Conection with the Graph Laplacian**:
As we are approximating a manifold with a graph, we need a suitable measure of smoothness for functions defined on the graph. It turns out that many of the concepts in the previous section have parallels in graph theory (e.g., see [3]). Let $G = (V, E)$ be a weighted graph on $n$ vertices. We assume that the vertices are

numbered and use the notation $i \sim j$ for adjacent vertices $i$ and $j$. The graph Laplacian of $G$ is defined as $L = D - W$, where $W$ is the weight matrix and $D$ is a diagonal matrix, $D_{ii} = \sum_j W_{ji}$. $L$ can be thought of as an operator on functions defined on vertices of the graph. It is not hard to see that $L$ is a self-adjoint positive semidefinite operator. By the (finite dimensional) spectral theorem any function on $G$ can be decomposed as a sum of eigenfunctions of $L$. If we think of $G$ as a model for the manifold $\mathcal{M}$ it is reasonable to assume that a function on $G$ is smooth if it does not change too much between nearby points. If $\mathbf{f} = (f_1, \ldots, f_n)$ is a function on $G$, then we can formalize that intuition by defining the smoothness functional $\mathcal{S}_G(\mathbf{f}) = \sum_{i \sim j} W_{ij}(f_i - f_j)^2$. It is not hard to show that $\mathcal{S}_G(\mathbf{f}) = \mathbf{f}\, L\, \mathbf{f}^T = \langle \mathbf{f}, L\mathbf{f} \rangle_G = \sum_{i=1}^{n} \lambda_i \langle \mathbf{f}, \mathbf{e}_i \rangle_G$ which is the discrete analogue of the integration by parts from the previous section. The inner product here is the usual Euclidean inner product on the vector space with coordinates indexed by the vertices of $G$, $\mathbf{e}_i$ are normalized eigenvectors of $L$, $L\mathbf{e}_i = \lambda_i \mathbf{e}_i$, $\|\mathbf{e}_i\| = 1$. All eigenvalues are non-negative and the eigenfunctions corresponding to the smaller eigenvalues can be thought as "more smooth". The smallest eigenvalue $\lambda_1 = 0$ corresponds to the constant eigenvector $\mathbf{e}_1$.

## 5 Experimental Results

### 5.1 Handwritten Digit Recognition

We apply our techniques to the problem of optical character recognition. We use the popular MNIST dataset which contains 28x28 grayscale images of handwritten digits.[1] We use the 60000 image training set for our experiments. For all experiments we use 8 nearest neighbours to compute the adjacency matrix. The adjacency matrices are very sparse which makes solving eigenvector problems for matrices as big as 60000 by 60000 possible. For a particular trial, we fix the number of labeled examples we wish to use. A random subset of the 60000 images is used with labels to form the labeled set $L$. The rest of the images are used without labels to form the unlabeled data $U$. The classification results (for $U$) are averaged over 20 different random draws for $L$. Shown in fig. 2 is a summary plot of classification accuracy on the unlabeled set comparing the nearest neighbors baseline with our algorithm that retains the number of eigenvectors by following taking it to be 20% of the number of labeled points. The improvements over the base line are significant, sometimes exceeding 70% depending on the number of labeled and unlabeled examples. With only 100 labeled examples (and 59900 unlabeled examples), the Laplacian classifier does nearly as well as the nearest neighbor classifier with 5000 labeled examples. Similarly, with 500/59500 labeled/unlabeled examples, it does slightly better than the nearest neighbor base line using 20000 labeled examples By comparing the results for the total 60000 point data set, and 10000 and 1000 subsets we see that adding unlabeled data consistently improves classification accuracy. When almost all of the data is labeled, the performance of our classifier is close to that of $k$-NN. It is not particularly surprising as our method uses the nearest neighbor information.

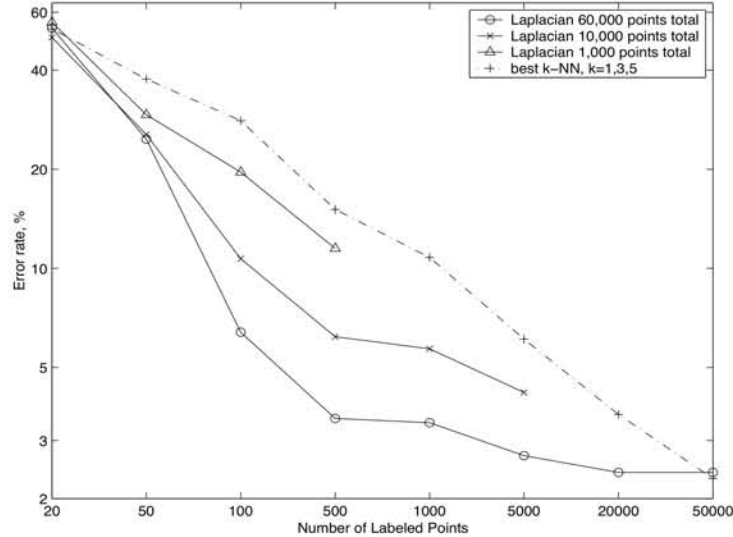

Figure 2: MNIST data set. Percentage error rates for different numbers of labeled and unlabeled points compared to best k-NN base line.

## 5.2 Text Classification

The second application is text classification using the popular 20 Newsgroups data set. This data set contains approximately 1000 postings from each of 20 different newsgroups. Given an article, the problem is to determine to which newsgroup it was posted. We tokenize the articles using the software package Rainbow written by Andrew McCallum. We use a "stop-list" of 500 most common words to be excluded and also exclude headers, which among other things contain the correct identification of the newsgroup. Each document is then represented by the counts of the most frequent 6000 words normalized to sum to 1. Documents with 0 total count are removed, thus leaving us with 19935 vectors in a 6000-dimensional space. We follow the same procedure as with the MNIST digit data above. A random subset of a fixed size is taken with labels to form $L$. The rest of the dataset is considered to be $U$. We average the results over 20 random splits[2]. As with the digits, we take the number of nearest neighbors for the algorithm to be 8. In fig. 3 we summarize the results by taking 19935, 2000 and 600 total points respectively and calculating the error rate for different numbers of labeled points. The number of eigenvectors used is always 20% of the number of labeled points. We see that having more unlabeled points improves the classification error in most cases although when there are very few labeled points, the differences are small.

## Footnotes

[1]We use the first 100 principal components of the set of all images to represent each image as a 100 dimensional vector.

[2]In the case of 2000 eigenvectors we take just 10 random splits since the computations are rather time-consuming.

## References

[1] M. Belkin, P. Niyogi, *Laplacian Eigenmaps for Dimensionality Reduction and Data Representation*, Technical Report, TR-2002-01, Department of Computer Science, The University of Chicago, 2002.

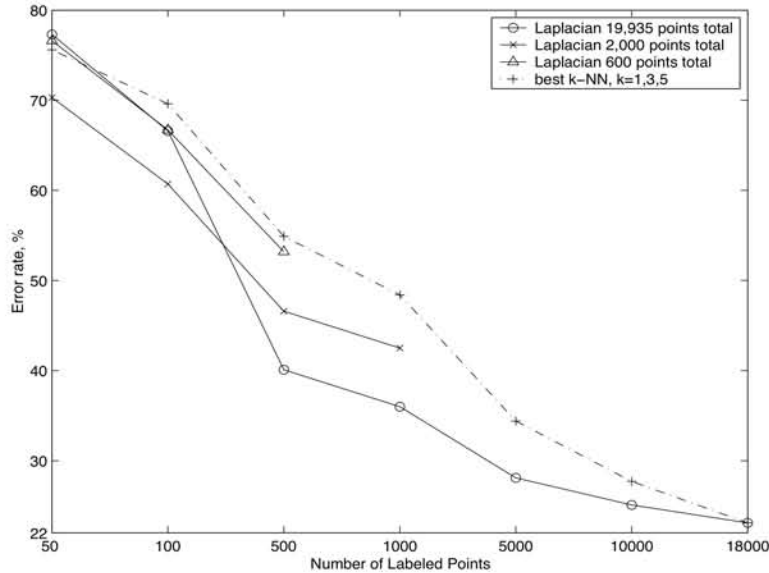

Figure 3: 20 Newsgroups data set. Error rates for different numbers of labeled and unlabeled points compared to best *k*-NN baseline.

[2] A. Blum, S. Chawla, *Learning from Labeled and Unlabeled Data using Graph Mincuts*, ICML, 2001,

[3] Fan R. K. Chung, *Spectral Graph Theory*, Regional Conference Series in Mathematics, number 92, 1997

[4] K. Nigam, A.K. McCallum, S. Thrun, T. Mitchell, *Text Classification from Labeled in Unlabeled Data*, Machine Learning 39(2/3), 2000,

[5] S. Rosenberg, *The Laplacian on a Riemmannian Manifold*, Cambridge University Press, 1997,

[6] Sam T. Roweis, Lawrence K. Saul, *Nonlinear Dimensionality Reduction by Locally Linear Embedding*, Science, vol 290, 22 December 2000,

[7] Martin Szummer, Tommi Jaakkola, *Partially labeled classification with Markov random walks*, Neural Information Processing Systems (NIPS) 2001, vol 14.,

[8] Joshua B. Tenenbaum, Vin de Silva, John C. Langford, *A Global Geometric Framework for Nonlinear Dimensionality Reduction*, Science, Vol 290, 22 December 2000,
